# Denoising and untangling graphs using degree priors

**Quaid D Morris, Brendan J Frey, and Christopher J Paige**
University of Toronto
Electrical and Computer Engineering
10 King's College Road, Toronto, Ontario, M5S 3G4
Canada
{quaid, frey}@psi.utoronto.ca, paige@uhnres.utoronto.ca

## Abstract

This paper addresses the problem of untangling hidden graphs from a set of noisy detections of undirected edges. We present a model of the generation of the observed graph that includes degree-based structure priors on the hidden graphs. Exact inference in the model is intractable; we present an efficient approximate inference algorithm to compute edge appearance posteriors. We evaluate our model and algorithm on a biological graph inference problem.

## 1 Introduction and motivation

The inference of hidden graphs from noisy edge appearance data is an important problem with obvious practical application. For example, biologists are currently building networks of all the physical protein-protein interactions (PPI) that occur in particular organisms. The importance of this enterprise is commensurate with its scale: a completed network would be as valuable as a completed genome sequence, and because each organism contains thousands of different types of proteins, there are millions of possible types of interactions. However, scalable experimental methods for detecting interactions are noisy, generating many false detections. Motivated by this application, we formulate the general problem of inferring hidden graphs as probabilistic inference in a graphical model, and we introduce an efficient algorithm that approximates the posterior probability that an edge is present.

In our model, a set of hidden, constituent graphs are combined to generate the observed graph. Each hidden graph is independently sampled from a prior on graph structure. The combination mechanism acts independently on each edge but can be either stochastic or deterministic. Figure 1 shows an example of our generative model. Typically one of the hidden graphs represents the graph of interest (the *true* graph), the others represent different types of observation noise. Independent edge noise may also be added by the combination mechanism. We use probabilistic inference to compute a likely decomposition of the observed graph into its constituent parts. This process is deemed "untangling". We use the term "denoising" to refer to the special case where the edge noise is independent. In denoising there is a single hidden graph, the true graph, and all edge noise in the observed graph is due

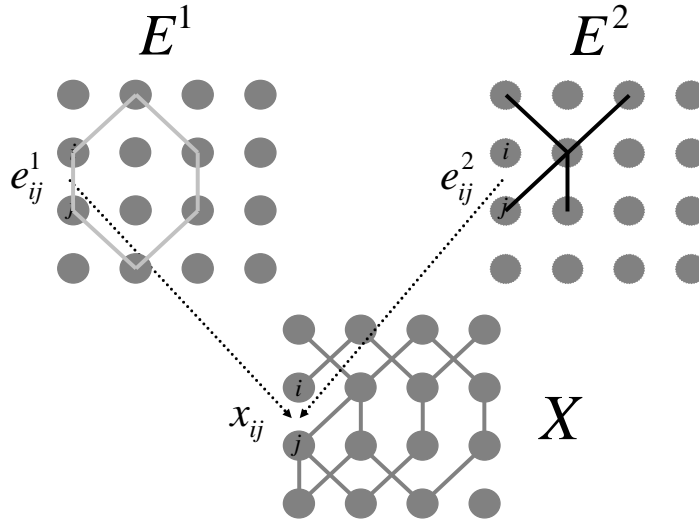

**Figure 1**: Illustrative generative model example. Figure shows an example where an observed graph, $X$, is a noisy composition of two constituent graphs, $E^1$ and $E^2$. All graphs share the same vertex set, so each can be represented by a symmetric matrix of random binary variables (i.e., an adjacency matrix). This generative model is designed to solve a toy counter-espionage problem. The vertices represent suspects and each edge in $X$ represents an observed call between two suspects. The graph $X$ reflects zero or more spy rings (represented by $E^1$), telemarketing calls (represented by $E^2$), social calls (independent edge noise), and lost call records (more independent edge noise). The task is to locate any spy rings hidden in $X$. We model the distribution of spy ring graphs using a prior, $P(E^1)$, that has support only on graphs where all vertices have degree of either 2 (i.e., are in the ring) or 0 (i.e., are not). Graphs of telemarketing call patterns are represented using a prior, $P(E^2)$, under which all nodes have degrees of $> 3$ (i.e., are telemarketers), 1 (i.e., are telemarketees), or 0 (i.e., are neither). The displayed hidden graphs are one likely untangling of $X$.

to the combination mechanism.

Prior distributions over graphs can be specified in various ways, but our choice is motivated by problems we want to solve, and by a view to deriving an efficient inference algorithm. One compact representation of a distribution over graphs consists of specifying a distribution over vertex degrees, and assuming that graphs that have the same vertex degrees are equiprobable. Such a prior can model quite rich distributions over graphs. These degree-based structure priors are natural representations of graph structure; many classes of real-world networks have a characteristic functional form associated with their degree distributions [1], and sometimes this form can be predicted using knowledge about the domain (see, e.g., [2]) or detected empirically (see, e.g., [3, 4]). As such, our model incorporates degree-based structure priors.

Though exact inference in our model is intractable in general, we present an efficient algorithm for approximate inference for arbitrary degree distributions. We evaluate our model and algorithm using the real-world example of untangling yeast protein-protein interaction networks.

## 2 A model of noisy and tangled graphs

For degree-based structure priors, inference consists of searching over vertex degrees and edge instantiations, while comparing each edge with its noisy observation and enforcing the constraint that the number of edges connected to every vertex must equal the degree of the vertex. Our formulation of the problem in this way is inspired by the success of the sum-product algorithm (loopy belief propagation) for solving similar formulations of problems in error-correcting decoding [6, 7], phase unwrapping [8], and random satisfiability [9]. For example, in error-correcting decoding, inference consists of searching over configurations of codeword bits, while comparing each bit with its noisy observation and enforcing parity-check constraints on subsets of bits [10].

For a graph on a set of $N$ vertices, $e_{ij}$ is a variable that indicates the presence of an edge connecting vertices $i$ and $j$: $e_{ij} = 1$ if there is an edge, and $e_{ij} = 0$ otherwise. We assume the vertex set is fixed, so each graph is specified by an adjacency matrix, $E = \{e_{ij}\}_{i,j=1}^{N}$. The degree of vertex $i$ is denoted by $d_i$ and the degree set by $D = \{d_i\}_{i=1}^{N}$. The observations are given by a noisy adjacency matrix, $X = \{x_{ij}\}_{i,j=1}^{N}$. Generally, edges can be directed, but in this paper we focus on undirected graphs, so $e_{ij} = e_{ji}$ and $x_{ij} = x_{ji}$.

Assuming the observation noise is independent for different edges, the joint distribution is

$$P(X, E, D) = P(X|E)P(E, D) = \left( \prod_{j \geq i} P(x_{ij}|e_{ij}) \right) P(E, D).$$

$P(x_{ij}|e_{ij})$ models the edge observation noise. We use an undirected model for the joint distribution over edges and degrees, $P(E, D)$, where the prior distribution over $d_i$ is determined by a non-negative potential $f_i(d_i)$. Assuming graphs that have the same vertex degrees are equiprobable, we have

$$P(E, D) \propto \prod_{i} \left( f_i(d_i) I(d_i, \sum_{j=1}^{N} e_{ij}) \right),$$

where $I(a, b) = 1$ if $a = b$, and $I(a, b) = 0$ if $a \neq b$. The term $I(d_i, \sum_j e_{ij})$ ensures that the number of edges connected to vertex $i$ is equal to $d_i$. It is straightforward to show that the marginal distribution over $d_i$ is $P(d_i) \propto f_i(d_i) \sum_{D \backslash d_i} \left( n_D \prod_{j \neq i} f_j(d_j) \right)$, where $n_D$ is the number of graphs with degrees $D$ and the sum is over all degree variables except $d_i$. The potentials, $f_i$, can be estimated from a given degree prior using Markov chain Monte Carlo; or, as an approximation, they can be set to an empirical degree distribution obtained from noise-free graphs.

Fig 2a shows the factor graph [11] for the above model. Each filled square corresponds to a term in the factorization of the joint distribution and the square is connected to all variables on which the term depends. Factor graphs are graphical models that unify the properties of Bayesian networks and Markov random fields [12]. Many inference algorithms, including the sum-product algorithm (a.k.a. loopy belief propagation), are more easily derived using factor graphs than Bayesian networks or Markov random fields. We describe the sum-product algorithm for our model in section 3.

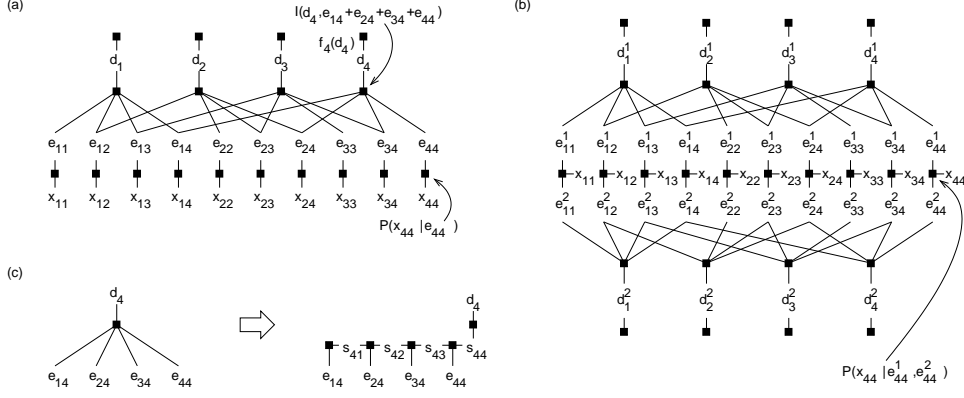

**Figure 2**: (a) A factor graph that describes a distribution over graphs with vertex degrees $d_i$, binary edge indicator variables $e_{ij}$, and noisy edge observations $x_{ij}$. The indicator function $I(d_i, \sum_j e_{ij})$ enforces the constraint that the sum of the binary edge indicator variables for vertex $i$ must equal the degree of vertex $i$. (b) A factor graph that explains noisy observed edges as a combination of two constituent graphs, with edge indicator variables $e_{ij}^1$ and $e_{ij}^2$. (c) The constraint $I(d_i, \sum_j e_{ij})$ can be implemented using a chain with state variables, which leads to an exponentially faster message-passing algorithm.

## 2.1 Combining multiple graphs

The above model is suitable when we want to infer a graph that matches a degree prior, assuming the edge observation noise is independent. A more challenging goal, with practical application, is to infer multiple hidden graphs that combine to explain the observed edge data. In section 4, we show how priors over multiple hidden graphs can be be used to infer protein-protein interactions.

When there are $H$ hidden graphs, each constituent graph is specified by a set of edges on the same set of $N$ common vertices. For the degree variables and edge variables, we use a superscript to indicate which hidden graph the variable is used to describe. Assuming the graphs are independent, the joint distribution over the observed edge data $X$, and the edge variables and degree variables for the hidden graphs, $E^1, D^1, \ldots, E^H, D^H$, is

$$P(X, E^1, D^1, \ldots, E^H, D^H) = \left( \prod_{j \geq i} P(x_{ij}|e_{ij}^1, \ldots, e_{ij}^H) \right) \prod_{h=1}^{H} P(E^h, D^h), \quad (1)$$

where for each hidden graph, $P(E^h, D^h)$ is modeled as described above. Here, the likelihood $P(x_{ij}|e_{ij}^1, \ldots, e_{ij}^H)$ describes how the edges in the hidden graphs combine to model the observed edge. Figure 2b shows the factor graph for this model.

## 3 Probabilistic inference of constituent graphs

Exact probabilistic inference in the above models is intractable, here we introduce an approximate inference algorithm that consists of applying the sum-product algorithm, while ignoring cycles in the factor graph. Although the sum-product algorithm has been used to obtain excellent results on several problems [6, 7, 13, 14, 8, 9], we have found that the algorithm works best when the model consists of uncertain observations of variables that are subject to a large number of hard constraints. Thus the formulation of the model described above.

Conceptually, our inference algorithm is a straight-forward application of the sum-product algorithm, c.f. [15], where messages are passed along edges in the factor graph iteratively, and then combined at variables to obtain estimates of posterior probabilities. However, direct implementation of the message-passing updates will lead to an intractable algorithm. In particular, direct implementation of the update for the message sent from function $I(d_i, \sum_j e_{ij})$ to edge variable $e_{ik}$ takes a number of scalar operations that is *exponential* in the number of vertices. Fortunately there exists a more efficient way to compute these messages.

## 3.1 Efficiently summing over edge configurations

The function $I(d_i, \sum_j e_{ij})$ ensures that the number of edges connected to vertex $i$ is equal to $d_i$. Passing messages through this function requires summing over all edge configurations that correspond to each possible degree, $d_i$, and summing over $d_i$. Specifically, the message, $\mu_{I_i \to e_{ik}}(e_{ik})$, sent from function $I(d_i, \sum_j e_{ij})$ to edge variable $e_{ik}$ is given by

$$\sum_{d_i} \sum_{\{e_{ij}|\ j=1,...,N,\ j \neq k\}} \left( I(d_i, \sum_j e_{ij}) \prod_{j \neq k} \mu_{e_{ij} \to I_i}(e_{ij}) \right),$$

where $\mu_{e_{ij} \to I_i}(e_{ij})$ is the message sent from $e_{ij}$ to function $I(d_i, \sum_j e_{ij})$.

The sum over $\{e_{ij}|\ j = 1, \ldots, N,\ j \neq k\}$ contains $2^{N-1}$ terms, so direct computation is intractable. However, for a maximum degree of $d_{\max}$, *all* messages departing from the function $I(d_i, \sum_j e_{ij})$ can be computed using order $d_{\max}N$ binary scalar operations, by introducing integer state variables $s_{ij}$. We define $s_{ij} = \sum_{n \leq j} e_{in}$ and note that, by recursion, $s_{ij} = s_{ij-1} + e_{ij}$, where $s_{i0} = 0$ and $0 \leq s_{ij} \leq d_{\max}$. This recursive expression enables us to write the high-complexity constraint as the sum of a product of low-complexity constraints,

$$I(d_i, \sum_j e_{ij}) = \sum_{\{s_{ij}|\ j=1,...,N\}} I(s_{i1}, e_{i1}) \left( \prod_{j=2}^{N} I(s_{ij}, s_{ij-1} + e_{ij}) \right) I(d_i, s_{iN}).$$

This summation can be performed using the forward-backward algorithm. In the factor graph, the summation can be implemented by replacing the function $I(d_i, \sum_j e_{ij})$ with a chain of lower-complexity functions, connected as shown in Fig. 2c. The function vertex (filled square) on the far left corresponds to $I(s_{i1}, e_{i1})$ and the function vertex in the upper right corresponds to $I(d_i, s_{iN})$. So, messages can be passed through each constraint function $I(d_i, \sum_j e_{ij})$ in an efficient manner, by performing a single forward-backward pass in the corresponding chain.

## 4  Results

We evaluate our model using yeast protein-protein interaction (PPI) data compiled by [16]. These data include eight sets of putative, but noisy, interactions derived from various sources, and one gold-standard set of interactions detected by reliable experiments.

Using the $\sim 6300$ yeast proteins as vertices, we represent the eight sets of putative interactions using adjacency matrices $\{Y^m\}_{m=1}^8$ where $y_{ij}^m = 1$ if and only if putative interaction dataset $m$ contains an interaction between proteins $i$ and $j$. We similarly use $Y^{\text{gold}}$ to represent the gold-standard interactions.

We construct an observed graph, $X$, by setting $x_{ij} = \max_m y_{ij}^m$ for all $i$ and $j$, thus the observed edge set is the union of all the putative edge sets. We test our model

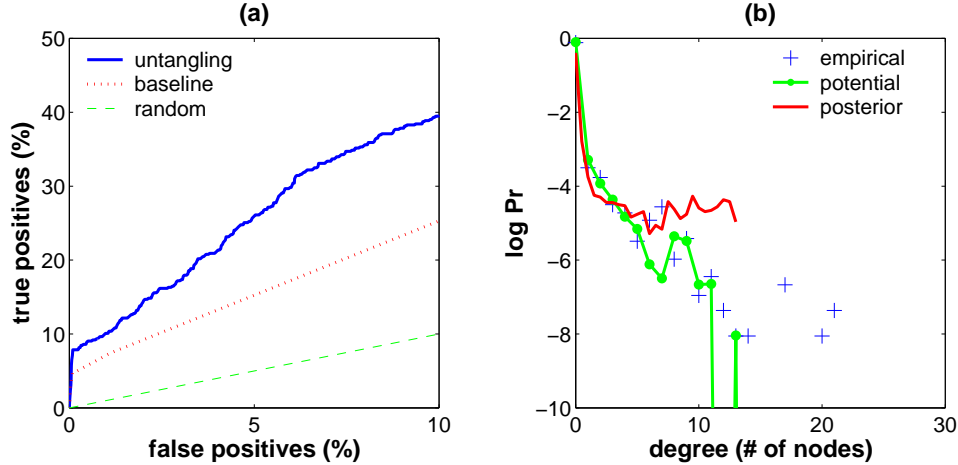

**Figure 3**: Protein-protein interaction network untangling results. (a) ROC curves measuring performance of predicting $e^1_{ij}$ when $x_{ij} = 1$. (b) Degree distributions. Compares the empirical degree distribution of the test set subgraph of $E^1$ to the degree potential $f^1$ estimated on the training set subgraph of $E^1$ and to the distribution of $d_i = \sum_j p_{ij}$ where $p_{ij} = \hat{P}(e^1_{ij} = 1|X)$ is estimated by untangling.

on the task of discerning which of the edges in $X$ are also in $Y^{\text{gold}}$. We formalize this problem as that of decomposing $X$ into two constituent graphs $E^1$ and $E^2$, the true and the noise graphs respectively, such that $e^1_{ij} = x_{ij}y^{\text{gold}}_{ij}$ and $e^2_{ij} = x_{ij} - e^1_{ij}$.

We use a training set to fit our model parameters and then measure task performance on a test set. The training set contains a randomly selected half of the $\sim 6300$ yeast proteins, and the subgraphs of $E^1$, $E^2$, and $X$ restricted to those vertices. The test contains the other half of the proteins and the corresponding subgraphs. Note that interactions connecting test set proteins to training set proteins (and vice versa) are ignored.

We fit three sets of parameters: a set of Naive Bayes parameters that define a set of edge-specific likelihood functions, $P_{ij}(x_{ij}|e^1_{ij}, e^2_{ij})$, one degree potential, $f^1$, which is the same for every vertex in $E_1$ and defines the prior $P(E^1)$, and a second, $f^2$, that similarly defines the prior $P(E^2)$.

The likelihood functions, $P_{ij}$, are used to both assign likelihoods and enforce problem constraints. Given our problem definition, if $x_{ij} = 0$ then $e^1_{ij} = e^2_{ij} = 0$, otherwise $x_{ij} = 1$ and $e^1_{ij} = 1 - e^2_{ij}$. We enforce the former constraint by setting $P_{ij}(x_{ij} = 0|e^1_{ij}, e^2_{ij}) = (1 - e^1_{ij})(1 - e^2_{ij})$, and the latter by setting $P_{ij}(x_{ij} = 1|e^1_{ij}, e^2_{ij}) = 0$ whenever $e^1_{ij} = e^2_{ij}$. This construction of $P_{ij}$ simplifies the calculation of the $\mu_{P_{ij} \to e^h_{ij}}$ messages and improves the computational efficiency of inference because when $x_{ij} = 0$, we need never update messages to and from variables $e^1_{ij}$ and $e^2_{ij}$. We complete the specification of $P_{ij}(x_{ij} = 1|e^1_{ij}, e^2_{ij})$ as follows:

$$P_{ij}(x_{ij} = 1|e^1_{ij}, e^2_{ij}) = \begin{cases} \theta_m^{y^m_{ij}}(1 - \theta_m)^{1-y^m_{ij}}, \text{if } e^1_{ij} = 1 \text{ and } e^2_{ij} = 0, \\ \psi_m^{y^m_{ij}}(1 - \psi_m)^{1-y^m_{ij}}, \text{if } e^1_{ij} = 0 \text{ and } e^2_{ij} = 1. \end{cases}$$

where $\{\theta_m\}$ and $\{\psi_m\}$ are naive Bayes parameters, $\theta_m = \sum_{i,j} y^m_{ij} e^1_{ij} / \sum_{i,j} e^1_{ij}$ and

$\psi_m = \sum_{i,j} y_{ij}^m e_{ij}^2 / \sum_{i,j} e_{ij}^2$, respectively.

The degree potentials $f^1(d)$ and $f^2(d)$ are kernel density estimates fit to the degree distribution of the training set subgraphs of $E^1$ and $E^2$, respectively. We use Gaussian kernels and set the width parameter (standard deviation) $\sigma$ using leave-one-out cross-validation to maximize the total log density of the held-out datapoints. Each datapoint is the degree of a single vertex. Both degree potentials closely followed the training set empirical degree distributions.

Untangling was done on the test set subgraph of $X$. We initially set the $\mu_{P_{ij} \to e_{ij}^1}$ messages equal to the likelihood function $P_{ij}$ and we randomly initialized the $\mu_{I_j^1 \to e_{ij}^1}$ messages with samples from a normal distribution with mean 0 and variance 0.01. We then performed 40 iterations of the following message update order: $\mu_{e_{ij}^1 \to I_j^1}, \mu_{I_j^1 \to e_{ij}^1}, \mu_{e_{ij}^1 \to P_{ij}}, \mu_{P_{ij} \to e_{ij}^2}, \mu_{e_{ij}^2 \to I_j^2}, \mu_{I_j^2 \to e_{ij}^2}, \mu_{e_{ij}^2 \to P_{ij}}, \mu_{P_{ij} \to e_{ij}^1}$.

We evaluated our untangling algorithm using an ROC curve by comparing the actual test set subgraph of $E^1$ to posterior marginal probabilities, $\hat{P}(e_{ij}^1 = 1|X)$, estimated by our sum-product algorithm. Note that because the true interaction network is sparse (less than 0.2% of the $1.8 \times 10^7$ possible interactions are likely present [16]) and, in this case, true positive predictions are of greater biological interest than true negative predictions, we focus on low false positive rate portions of the ROC curve.

Figure 3a compares the performance of a classifier for $e_{ij}^1$ based on thresholding $\hat{P}(e_{ij} = 1|X)$ to a baseline method based on thresholding the likelihood functions, $P_{ij}(x_{ij} = 1|e_{ij}^1 = 1, e_{ij}^2 = 0)$. Note because $e_{ij}^1 = 0$ whenever $x_{ij} = 0$, we exclude the $x_{ij} = 0$ cases from our performance evaluation. The ROC curve shows that for the same low false positive rate, untangling produces $50\% - 100\%$ more true positives than the baseline method.

Figure 3b shows that the degree potential, the true degree distribution, and the predicted degree distribution are all comparable. The slight overprediction of the true degree distribution may result because the degree potential $f^1$ that defines $P(E^1)$ is not equal to the expected degree distribution of graphs sampled from the distribution $P(E^1)$.

## 5  Summary and Related Work

Related work includes other algorithms for structure-based graph denoising [17, 18]. These algorithms use structural properties of the observed graph to score edges and rely on the true graph having a surprisingly large number of three (or four) edge cycles compared to the noise graph. In contrast, we place graph generation in a probabilistic framework; our algorithm computes structural fit in the hidden graph, where this computation is not affected by the noise graph(s); and we allow for multiple sources of observation noise, each with its own structural properties.

After submitting this paper to the NIPS conference, we discovered [19], in which a degree-based graph structure prior is used to denoise (but not untangle) observed graphs. This paper addresses denoising in directed graphs as well as undirected graphs, however, the prior that they use is not amenable to deriving an efficient sum-product algorithm. Instead, they use Markov Chain Monte Carlo to do approximate inference in a hidden graph containing 40 vertices. It is not clear how well this approach scales to the $\sim 3000$ vertex graphs that we are using.

In summary, the contributions of the work described in this paper include: a general

formulation of the problem of graph untangling as inference in a factor graph; an efficient approximate inference algorithm for a rich class of degree-based structure priors; and a set of reliability scores (i.e., edge posteriors) for interactions from a current version of the yeast protein-protein interaction network.

# References

[1] A L Barabasi and R Albert. Emergence of scaling in random networks. *Science*, 286(5439), October 1999.

[2] A Rzhetsky and S M Gomez. Birth of scale-free molecular networks and the number of distinct dna and protein domains per genome. *Bioinformatics*, pages 988–96, 2001.

[3] M Faloutsos, P Faloutsos, and C Faloutsos. On power-law relationships of the Internet topology. *Computer Communications Review*, 29, 1999.

[4] Hawoong Jeong, B Tombor, Réka Albert, Z N Oltvai, and Albert-László Barabási. The large-scale organization of metabolic networks. *Nature*, 407, October 2000.

[5] J. Pearl. *Probabilistic Reasoning in Intelligent Systems*. Morgan Kaufmann, San Mateo CA., 1988.

[6] D. J. C. MacKay and R. M. Neal. Near Shannon limit performance of low density parity check codes. *Electronics Letters*, 32(18):1645–1646, August 1996. Reprinted in *Electronics Letters*, vol. 33, March 1997, 457–458.

[7] B. J. Frey and F. R. Kschischang. Probability propagation and iterative decoding. In *Proceedings of the 1996 Allerton Conference on Communication, Control and Computing*, 1996.

[8] B. J. Frey, R. Koetter, and N. Petrovic. Very loopy belief propagation for unwrapping phase images. In *2001 Conference on Advances in Neural Information Processing Systems, Volume 14*. MIT Press, 2002.

[9] M. Mézard, G. Parisi, and R. Zecchina. Analytic and algorithmic solution of random satisfiability problems. *Science*, 297:812–815, 2002.

[10] B. J. Frey and D. J. C. MacKay. Trellis-constrained codes. In *Proceedings of the $35^{th}$ Allerton Conference on Communication, Control and Computing 1997*, 1998.

[11] F. R. Kschischang, B. J. Frey, and H.-A. Loeliger. Factor graphs and the sum-product algorithm. *IEEE Transactions on Information Theory, Special Issue on Codes on Graphs and Iterative Algorithms*, 47(2):498–519, February 2001.

[12] B. J. Frey. Factor graphs: A unification of directed and undirected graphical models. University of Toronto Technical Report PSI-2003-02, 2003.

[13] Kevin P. Murphy, Yair Weiss, and Michael I. Jordan. Loopy belief propagation for approximate inference: An empirical study. In *Uncertainty in Artificial Intelligence 1999*. Stockholm, Sweden, 1999.

[14] W. Freeman and E. Pasztor. Learning low-level vision. In *Proceedings of the International Conference on Computer Vision*, pages 1182–1189, 1999.

[15] M. I. Jordan. *An Inroduction to Learning in Graphical Models*. 2004. In preparation.

[16] C von Mering et al. Comparative assessment of large-scale data sets of protein-protein interactions. *Nature*, 2002.

[17] R Saito, H Suzuki, and Y Hayashizaki. Construction of reliable protein-protein interaction networks with a new interaction generality measure. *Bioinformatics*, pages 756–63, 2003.

[18] D S Goldberg and F P Roth. Assessing experimentally derived interactions in a small world. *Proceedings of the National Academy of Science*, 2003.

[19] S M Gomez and A Rzhetsky. Towards the prediction of complete protein–protein interaction networks. In *Pacific Symposium on Biocomputing*, pages 413–24, 2002.
